# Algorithms for Better Representation and Faster Learning in Radial Basis Function Networks

Avijit   Saha[1]
**James D. Keeler**
Microelectronics and Computer Technology corporation
3500 West Balcones Center Drive
Austin, Tx 78759

## ABSTRACT

In this paper we present upper bounds for the learning rates for hybrid models that employ a combination of both self-organized and supervised learning, using radial basis functions to build receptive field representations  in  the hidden units. The learning performance in such networks with nearest neighbor heuristic can be improved upon by multiplying the individual receptive field widths by a suitable overlap factor. We present results indicating optimal values for such overlap factors. We also present a new algorithm for determining receptive field centers. This method negotiates more hidden units in the regions of the input space  as a function of the output and is conducive to better learning when the number of patterns (hidden units) is small.

## 1 INTRODUCTION

Functional approximation of experimental data originating from a continuous dynamical process is an important problem.  Data is usually available in the form of a set S consisting of {x,y} pairs, where x is a input vector and y is the corresponding output vector. In particular, we consider networks with a single layer of hidden units and the $j^{th}$ output unit computes $y_j = \Sigma f_\alpha R_\alpha \{ x_j, x_\alpha, \sigma_\alpha \}$, where, $y_j$ is the

[1] University of Texas at Austin, Dept. of ECE, Austin TX 78712

network output due to input $x_j$, $f_\alpha$ is the synaptic weight associated with the $\alpha^{th}$ hidden neuron and the $j^{th}$ output unit; $R_\alpha\{ x(t_j), x_\alpha, \sigma \}$ is the Radial Basis Function (RBF) response of the $\alpha^{th}$ hidden neuron. This technique of using a superposition of RBF for the purposes of approximation has been considered before by [Medgassy '58] and more recently by [Moody '88], [Casdagli '89] and [Poggio '89]. RBF networks are particularly attractive since such networks are potentially 1000 times faster than the ubiquitous backpropagation network for comparable error rates [Moody '88].

The essence of the network model we consider is described in [Moody '88]. A typical network that implements a receptive field response consists of a layer of linear input units, a layer of linear output units and an intermediate ( hidden ) layer of non-linear response units. Weights are associated with only the links connecting the hidden layer to the output layer. For the single output case the real valued functional mapping $f : R^n \rightarrow R$ is characterized by the following equations:

$$O(x_i) = \Sigma f_\alpha R_\alpha (x_i) \tag{1}$$

$$O(x_i) = \Sigma f_\alpha R_\alpha (x_i) / \Sigma R_\alpha (x_i) \tag{2}$$

$$R_\alpha(x_i) = e^{- ( |x_\alpha - x_i| / \sigma_\alpha )^2} \tag{3}$$

where $x_\alpha$ is a real valued vector associated with the $\alpha^{th}$ receptive field ( hidden ) unit and is of the same dimension as the input. The output can be normalized by the sum of the responses of the hidden units due to any input, and the expression for the output using normalized response function is presented in Equation 2. The $x_\alpha$ values the centers of the receptive field units and $\sigma_\alpha$ are their widths. Training in such networks can be performed in a two stage hybrid combination of independent processes. In the first stage, a clustering of the input data is performed. The objective of this clustering algorithm is to establish appropriate $x_\alpha$ values for each of the receptive field units such that the cluster points represent the input distribution in the best possible manner. We use competetive learning with the nearest neighbor heuristic as our clustering algorithm (Equation 5). The degree or quality of clustering achieved is quantified by the sum-square measure in Equation 4, which is the objective function we are trying to minimize in the clustering phase.

$$\text{TSS- KMEANS} = \Sigma ( x_{\alpha-closest} - x_i )^2 \tag{4}$$

$$x_{\alpha-closest} = x_{\alpha-closest} + \lambda ( x_i - x_{\alpha-closest} ) \tag{5}$$

After suitable cluster points ($x_\alpha$ values) are determined the next step is to determine

the $\sigma_\alpha$ or widths for each of the receptive fields. Once again we use the nearest neighbor heuristic where $\sigma_\alpha$ (the width of the $\alpha^{th}$ neuron) is set equal to the euclidian distance between $x_\alpha$ and its nearest neighbor. Once the receptive field centers $x_\alpha$ and the widths ($\sigma_\alpha$) are found, the receptive field responses can be calculated for any input using Equation 3. Finally, the $f_\alpha$ values or weights on links connecting the hidden layer units to the output are determined using the well-known gradient descent learning rule. Pseudo inverse methods are usually impractical in these problems. The rules for the objective function and weight update are given by equations 6 and 7.

$$E \quad = \quad \Sigma_i (O(x_i) - t_i)^2 \tag{6}$$

$$f_\alpha \quad = \quad f_\alpha + \eta \{(O(x_i) - t_i)\} R_\alpha (x_i) \tag{7}$$

where, $i$ is the number of input patterns, $x_i$ is the input vector and $t_i$ is the target output for the $i^{th}$ pattern.

## 2 LEARNING RATES

In this section we present an adaptive formulation for the network learning rate $\eta$ (Equation 7). Learning rates ($\eta$) in such networks that use gradient descent are usually chosen in an adhoc fashion. A conservative value for $\eta$ is usually sufficient. However, there are two problems with such an approach. If the learning rate is not small enough the TSS (Total Sums of Squares) measure can diverge to high values instead of decreasing. A very conservative estimate on the other hand will work with almost all sets of data but will unnecessarily slow down the learning process. The choice of learning rate is crucial, since for real-time or hardware implementations of such systems there is very little scope for interactive monitoring.

This problem is addressed by the Theorem 1. We present the proof for this theorem for the special case of a single output. In the gradient descent algorithm, weight updates can be performed after each presentation of the entire set of patterns (per epoch basis or after each pattern (per pattern basis); both cases are considered. Equation p.3 gives the upper bound for $\eta$ when updates are done on a per epoch basis. Only positive values of $\eta$ should be considered. Equations p.4 and p.5 gives the bounds for $\eta$ when updates are done on a per pattern basis without and with normalized response function respectively. We present some simulation results for the logistic map { $x(t+1) = r \, x(t) \, [1 - x(t)]$ } data in Figure 1. The plots are shown only for the normalized response case, and the learning rate was set to $\eta = \mu\{ (\Sigma R_\alpha)^2 / \Sigma (R_\alpha)^2 \}$. We used a fixed number of 20 hidden units, and r was set to 4.0. The network TSS did not diverge until $\mu$ was set arbitrarily close to 1.

This is shown in Figure 1 which indicates that, with the normalized response function, if the sum of squares of the hidden unit responses is nearly equal to the square of the sum of the responses, then a high effective learning rate ($\eta$) can be used.

**Theorem 1** : The TSS measure of a network will be decreasing in time, provided the learning rate $\eta$ does not exceed $\Sigma_i e_i \Sigma_\alpha E_\alpha R_{\alpha i} / \{ \Sigma_i (\Sigma_\alpha E_\alpha R_{\alpha i})^2 \}$ if the network is trained on a per epoch basis, and $1 / \Sigma_\alpha (R_{\alpha i})^2$ when updates are done on a per pattern basis. With normalized response function, the upper bound for the learning rate is $(\Sigma_\alpha R_{\alpha i})^2 / \Sigma_\alpha (R_{\alpha i})^2$. Note similar result of [Widrow 1985].

**Proof** :

$$TSS(t) \ = \ \Sigma_i (t_i - \Sigma_\alpha f_\alpha R_{\alpha i})^2 \qquad (p.1)$$

where N is the number of exemplars, and K is the number of receptive fields and $t_i$ is the $i^{th}$ target output.

$$TSS(t+1) \ = \ \Sigma_i (t_i - \Sigma_\alpha (f_\alpha + \Delta f_\alpha) R_{\alpha i})^2 \qquad (p.2)$$

$$\text{where, } \Delta f_\alpha \ = \ -\eta \ \frac{\delta (TSS(t)}{\delta f_\alpha}$$

$$= \ 2\eta \Sigma_i e_i R_{\alpha i}$$

$$= \ 2\eta E_\alpha \qquad \text{where, } E_\alpha \ = \ \Sigma_i e_i R_{\alpha i}$$

For stability, we impose the condition $TSS(t) - TSS(t+1) \geq 0$. From Eqns (p.1) and (p.2) above and substituting $\eta\, 2E_\alpha$ for $\Delta f_\alpha$, we have :

$$TSS(t) - TSS(t+1) \geq \Sigma_i e_i^2 - \Sigma_i (t_i - \Sigma_\alpha f_\alpha R_{\alpha i} - \Sigma_\alpha 2\eta E_\alpha R_{\alpha i})^2$$

Expanding the RHS of the above expression and substituting $e_i$ appropriately :

$$TSS(t) - TSS(t+1) \geq -4\eta \Sigma_i e_i \Sigma_\alpha E_\alpha R_{\alpha i} + 4\eta^2 \Sigma_i (\Sigma_\alpha E_\alpha R_{\alpha i})^2$$

$\therefore$ From the above inequality it follows that for stability in per epoch basis training, the upper bound for learning rate $\eta$ is given by :

$$\eta \ \leq \ \Sigma_i e_i \Sigma_\alpha E_\alpha R_{\alpha i} / \Sigma_i (\Sigma_\alpha E_\alpha R_{\alpha i})^2 \qquad (p.3)$$

If updates are done on a per pattern basis, then N = 1 and we drop the summation over N and the index i and we obtain the following bound :

$$\eta \ \leq \ 1 / \Sigma_\alpha (R_{\alpha i})^2. \qquad (p.4)$$

With normalized response function the upper bound for the learning rate is :

$$\eta \ \leq \ (\Sigma_\alpha R_{\alpha i})^2 / \Sigma_\alpha (R_{\alpha i})^2. \qquad ((p.5)$$

Q.E.D.

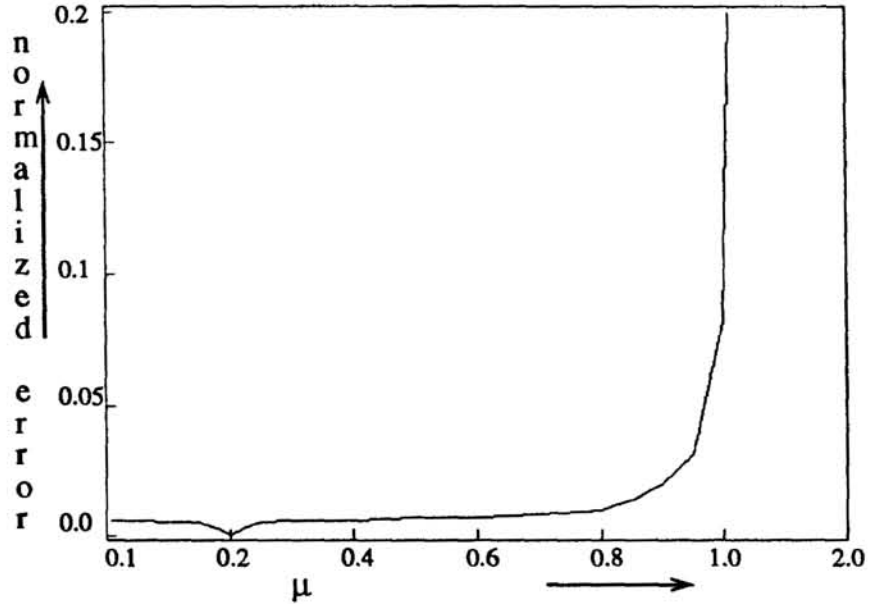

**Figure 1:** Normalized error vs. fraction ($\mu$ )
of maximum allowable learning rate

## 3 EFFECT OF WIDTH ($\sigma$) ON APPROXIMATION ERROR

In the nearest-neighbor heuristic $\sigma$   values of the hidden units are set equal to the
euclidian  distance  between its center  and  the  center of  its  nearest neighbor. This
method is preferred   mainly because it is computationally inexpensive. However, the
performance can be improved by increasing the overlap between nearby hidden unit
responses. This  is  done  by  multiplying  the  widths  obtained  with  the  nearest
neighbor heuristic by an overlap factor m as shown in Equation 3.1.

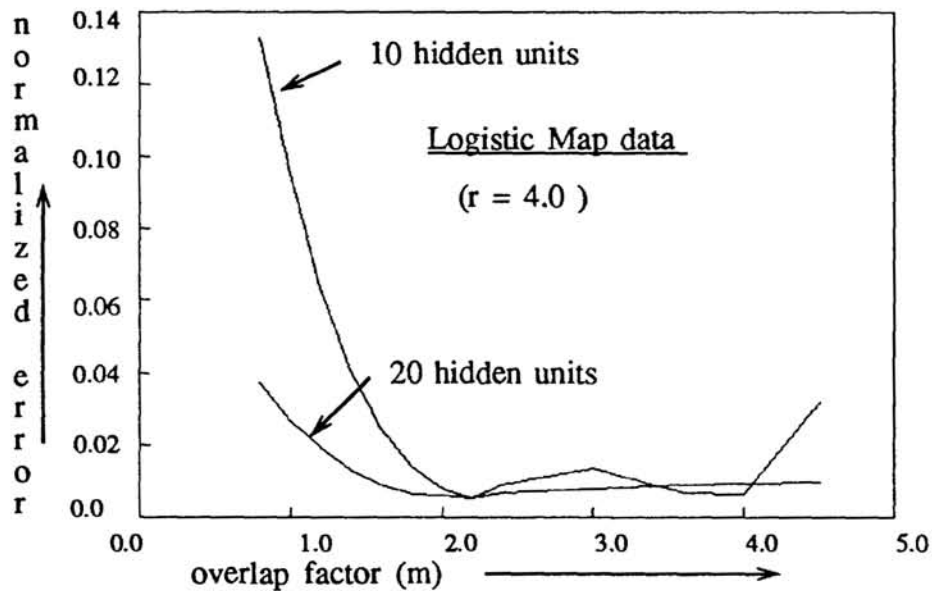

**Figure 2:** Normalized errors vs. overlap factor for the
logistic map.

$$\sigma_\alpha = m \cdot \| x_\alpha - x_{\alpha\text{-nearest}} \| \qquad (3.1)$$

and  $\| \cdot \|$  is the euclidian distance norm.

In Figures 2 and 3 we show the network performance ( normalized error )  as a function of m. In the logistic map case a value of r = 4.0 was used, predicting 1 timestep into the future; training set size was 10 times the number of hidden units and test set size was 70 patterns. The results for the Mackey-Glass data are with parameter values a = 0.1, b = 0.2, $\Delta$ = 6, D = 4. The number of training patterns was 10 times the number of hidden units and the normalized error was evaluated based on the presentation of 900 unseen patterns. For the Mackey-Glass data the optimal values were rather well-defined; whereas for the logistic map case we found that the optimal values were spread out over a range.

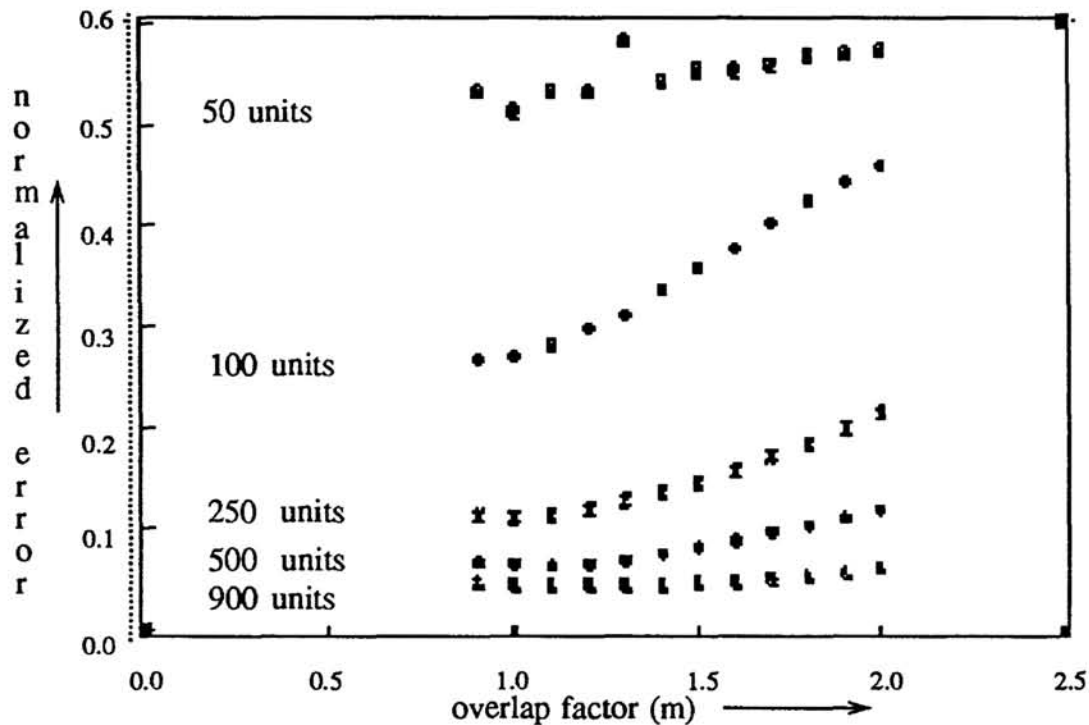

**Figure 2:**  Normalized errors vs. overlap factor for varying
number of hidden units, Mackey-Glass data.

## 4 EXTENDED METRIC CLUSTERING

In this method clustering is done in higher dimensions. In our experiments we set the initial K hidden unit center values based on the first K exemplars. The receptive fields are assigned vector values of dimensions determined by  the  size  of  the input and the output vectors. Each center value was set equal to the vector obtained by concatenating the input and the corresponding output. During the clustering phase the output $y_i$ is concatenated with the input $x_i$ and presented to the hidden layer.

This method finds cluster points in the (I+O)-dimensional space of the input-output map as defined by Equations 4.1, 4.2 and 4.3.

$$X_\alpha = <x_\alpha, y_\alpha> \tag{4.1}$$

$$X_i = <x_i, y_i> \tag{4.2}$$

$$X_{\alpha\text{-new}} = X_{\alpha\text{-old}} + \lambda(X_\alpha - X_i) \tag{4.3}$$

Once the cluster points or the centers are determined we disable the output field, and only the input field is used for computing the widths and receptive field responses. In Figure 3 we present a comparison of the performances of such a network with and without the enhanced metric clustering. Variable size networks of only Gaussian RBF units were used. The plots presented are for the Mackey-Glass data with the same parameter values used in [Farmer 88]. This method works significantly better when the number of hidden units is low.

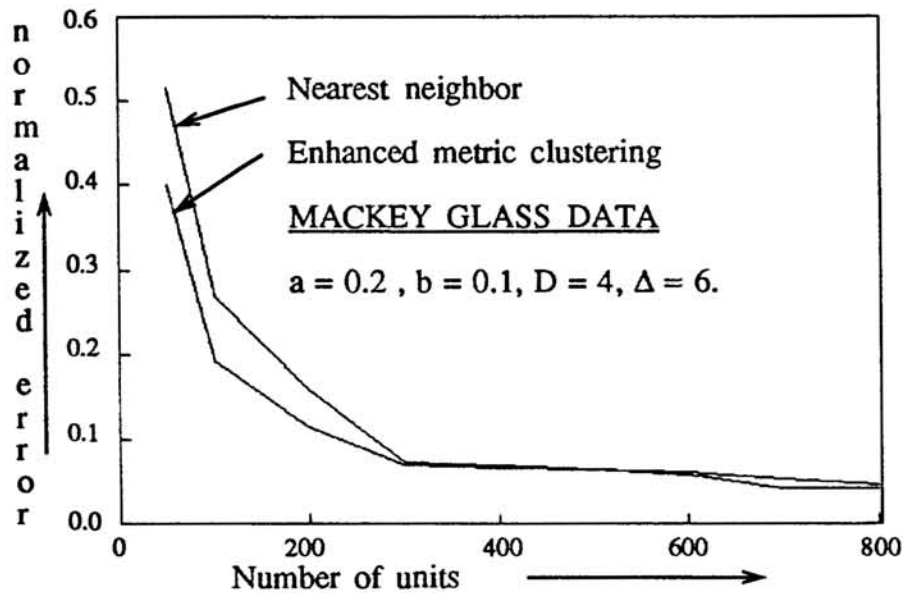

**Figure 3 :** Performance of enhanced metric clustering algorithmm.

## 5 CONCLUSIONS

One of the emerging application areas for neural network models is real time signal processing. For such applications and hardware implementations, adaptive methods for determining network parameters are essential. Our derivations for learning rates are important in such situations. We have presented results indicating that in RBF networks, performance can be improved by tuning the receptive field widths by some suitable overlap factor. We have presented an extended metric algorithm that negotiates hidden units based on added output information. We have observed more than 20% improvement in the normalized error measure when the number of training

patterns, and therefore the number of hidden units, used is reasonably small.

## References

M. Casdagli. (1989) "Nonlinear Prediction of Chaotic Time Series" Physica 35D, 335 -356.

D. J. Farmer and J. J. Sidorowich. (1988). "Exploiting Chaos to Predict the Future and Reduce Noise". Tech. Report No. LA-UR-88-901, Los Alamos National Laboratory.

John Moody and Christen Darken (1989)."Learning with Localised Receptive Fields". In: Eds: D. Touretzky, Hinton and Sejnowski: Proceedings of the 1988 Connectionist Models Summer School. Morgan Kaufmann Publishing, San Mateo, CA.

P. Medgassy. (1961) Decomposition of Superposition of Distribution Functions, Publishing house of the Hungarian Academy of Sciences, Budapest, 1961.

T. Poggio and F. Girosi. (1989). "A Theory of Networks for Approximation and Learning", A.I. Memo No. 1140, Massachusetts Institute of Technology.

B. Widrow and S. Stearns (1985). Adaptive Signal Processing. Prentice-Hall Inc., Englewood Cliffs, NJ, pp 49,102.
